# A Neural Network Classifier for the I1000 OCR Chip

**John C. Platt and Timothy P. Allen**
Synaptics, Inc.
2698 Orchard Parkway
San Jose, CA 95134
platt@synaptics.com, tpa@synaptics.com

## Abstract

This paper describes a neural network classifier for the I1000 chip, which optically reads the E13B font characters at the bottom of checks. The first layer of the neural network is a hardware linear classifier which recognizes the characters in this font. A second software neural layer is implemented on an inexpensive microprocessor to clean up the results of the first layer. The hardware linear classifier is mathematically specified using constraints and an optimization principle. The weights of the classifier are found using the active set method, similar to Vapnik's separating hyperplane algorithm. In 7.5 minutes of SPARC 2 time, the method solves for 1523 Lagrange multipliers, which is equivalent to training on a data set of approximately 128,000 examples. The resulting network performs quite well: when tested on a test set of 1500 real checks, it has a 99.995% character accuracy rate.

## 1   A BRIEF OVERVIEW OF THE I1000 CHIP

At Synaptics, we have created the I1000, an analog VLSI chip that, when combined with associated software, optically reads the E13B font from the bottom of checks. This E13B font is shown in figure 1. The overall architecture of the I1000 chip is shown in figure 2. The I1000 recognizes checks hand-swiped through a slot. A lens focuses the image of the bottom of the check onto the retina. The retina has circuitry which locates the vertical position of the characters on the check. The retina then sends an image vertically centered around a possible character to the classifier.

The classifier in the I1000 has a tough job. It must be very accurate and immune to noise and ink scribbles in the input. Therefore, we decided to use an integrated segmentation and recognition approach (Martin & Pittman, 1992)(Platt, et al., 1992). When the classifier produces a strong response, we know that a character is horizontally centered in the retina.

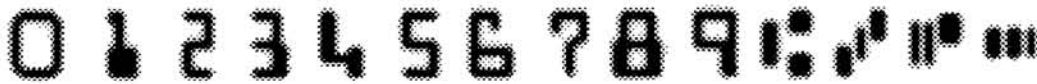

Figure 1: The E13B font, as seen by the I1000 chip

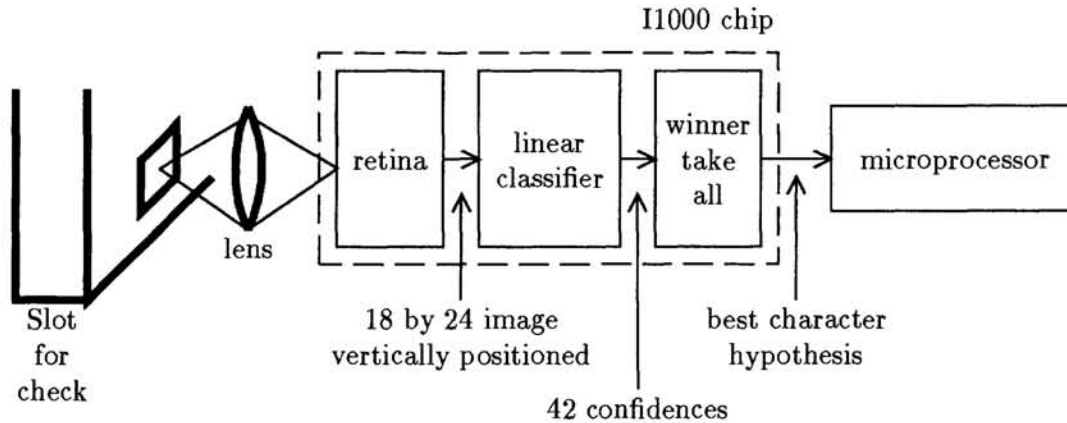

Figure 2: The overall architecture of the I1000 chip

We decided to use analog VLSI to minimize the silicon area of the classifier. Because of the analog implementation, we decided to use a linear template classifier, with fixed weights in silicon to minimize area. The weights are encoded as lengths of transistors acting as current sources. We trained the classifier using only the specification of the font, because we did not have the real E13B data at the time of classifier design. The design of the classifier is described in the next section.

As shown in figure 2, the input to the classifier is an 18 by 24 pixel image taken from the retina at a rate of 20 thousand frames per second. The templates in the classifier are 18 by 22 pixels. Each template is evaluated in three different vertical positions, to allow the retina to send a slightly vertically mis-aligned character. The output of the classifier is a set of 42 confidences, one for each of the 14 characters in the font in three different vertical positions. These confidences are fed to a winner-take-all circuit (Lazzaro, et al., 1989), which finds the confidence and the identity of the best character hypothesis.

## 2  SPECIFYING THE BEHAVIOR OF THE CLASSIFIER

Let us consider the training of one template corresponding to one of the characters in the font. The template takes a vector of pixels as input. For ease of analog implementation, the template is a linear neuron with no bias input:

$$O = \sum_i W_i I_i \qquad (1)$$

where $O$ is the output of the template, $W_i$ are the weights of the template, and $I_i$ are the input pixels of the template.

We will now mathematically express the training of the templates as three types of constraints on the weights of the template. The input vectors used by these constraints are the ideal characters taken from the specification of the font.

The first type of constraint on the template is that the output of the template should be above 1 when the character that corresponds to the template is centered

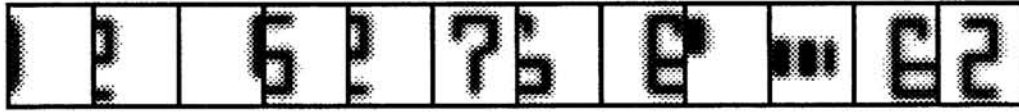

Figure 3: Examples of images from the bad set for the templates trained to detect the zero character. These images are E13B characters that have been horizontally and vertically offset from the center of the image. The black border around each of the characters shows the boundary of the input field. Notice the variety of horizontal and vertical shifts of the different characters.

in the horizontal field. Call the vector of pixels of this centered character $G_i$. This constraint is stated as:

$$\sum_i W_i G_i \geq 1 \qquad (2)$$

The second type of constraint on the template is to have an output much lower than 1 when incorrect or offset characters are applied to the template. We collect these incorrect and offset characters into a set of pixel vectors $\vec{B}^j$, which we call the "bad set." The constraint that the output of the template be lower than a constant $c$ for all of the vectors in the bad set is expressed as:

$$\sum_i W_i B_i^j \leq c \quad \forall j \qquad (3)$$

Together, constraints (2) and (3) permit use of a simple threshold to distinguish between a positive classifier response and a negative one.

The bad set contains examples of the correct character for the template that are horizontally offset by at least two pixels and vertically offset by up to one pixel. In addition, examples of all other characters are added to the bad set at every horizontal offset and with vertical offsets of up to one pixel (see figure 3). Vertically offset examples are added to make the classifier resistant to characters whose baselines are slightly mismatched.

The third type of constraint on the template requires that the output be invariant to the addition of a constant to all of the input pixels. This constraint makes the classifier immune to any changes in the background lighting level, $k$. This constraint is equivalent to requiring the sum of the weights to be zero:

$$\sum_i W_i(I_i + k) = \sum_i W_i I_i \quad \Rightarrow \quad \sum_i W_i = 0 \qquad (4)$$

Finally, an optimization principle is necessary to choose between all possible weight vectors that fulfill constraints (2), (3), and (4). We minimize the perturbation of the output of the template given uncorrelated random noise on the input. This optimization principle is similar to training on a large data set, instead of simply the ideal characters described by the specification. This optimization principle is equivalent to minimizing the sum of the square of the weights:

$$\min \sum_i W_i^2 \qquad (5)$$

Expressing the training of the classifier as a combination of constraints and an optimization principle allows us to compactly define its behavior. For example, the combination of constraints (3) and (4) allows the classifier to be immune to situations when two partial characters appear in the image at the same time. The confluence of two characters in the image can be described as:

$$I_i^{\text{overlap}} = k + B_i^l + B_i^r \qquad (6)$$

where $k$ is a background value and $B_i^l$ and $B_i^r$ are partial characters from the bad set that appears on the left side and right side of the image, respectively. The output of the template is then:

$$O^{\text{overlap}} = \sum_i W_i(k + B_i^l + B_i^r) = \sum_i W_i k + \sum_i W_i B_i^l + \sum_i W_i B_i^r < 2c \quad (7)$$

Constraints (3) and (4) thus limit the output of the neuron to less than $2c$ when two partial characters appear in the input. Therefore, we want $c$ to be less than 0.5. In order to get a 2:1 margin, we choose $c = 0.25$.

The classifier is trained only on individual partial characters instead of all possible combinations of partial characters. Therefore, we can specify the classifier using only 1523 constraints, instead of creating a training set of approximately 128,000 possible combinations of partial characters. Applying these constraints is therefore much faster than back-propagation on the entire data set.

Equations (2), (3) and (5) describe the optimization problem solved by Vapnik (Vapnik, 1982) for constructing a hyperplane that separates two classes. Vapnik solves this optimization problem by converting it into a dual space, where the inequality constraints become much simpler. However, we add the equality constraint (4), which does not allow us to directly use Vapnik's dual space method. To overcome this limitation, we use the active set method, which can fulfill any extra linear equality or inequality constraints. The active set method is described in the next section.

## 3   THE ACTIVE SET METHOD

Notice that constraints (2), (3), and (4) are all linear in $W_i$. Therefore, minimizing (5) with these constraints is simply quadratic programming with a mixture of equality and inequality constraints. This problem can be solved using the active set method from optimization theory (Gill, et al., 1981).

When the quadratic programming problem is solved, some of the inequality constraints and all of the equality constraints will be "active." In other words, the active constraints affect the solution as equality constraints. The system has "bumped into" these constraints. All other constraints will be inactive; they will not affect the solution.

Once we know which constraints are active, we can easily solve the quadratic minimization problem with equality constraints via Lagrange multipliers. The solution is a saddle point of the function:

$$\frac{1}{2}\sum_i W_i^2 + \sum_k \lambda_k(\sum_j A_{kj}W_j - C_k) \quad (8)$$

where $\lambda_k$ is the Lagrange multiplier of the $k$th active constraint, and $A_{kj}$ and $C_k$ are the linear and constant coefficients of the $k$th active constraint. For example, if constraint (2) is the $k$th active constraint, then $\vec{A}_k = \vec{G}$ and $C_k = 1$. The saddle point can be found via the set of linear equations:

$$W_i = -\sum_k \lambda_k A_{ki} \quad (9)$$

$$\lambda_k = -\sum_j(\sum_i A_{ji}A_{ki})^{-1}C_j \quad (10)$$

The active set method determines which inequality constraints belong in the active set by iteratively solving equation (10) above. At every step, one inequality constraint is either made active, or inactive. A constraint can be moved to the active

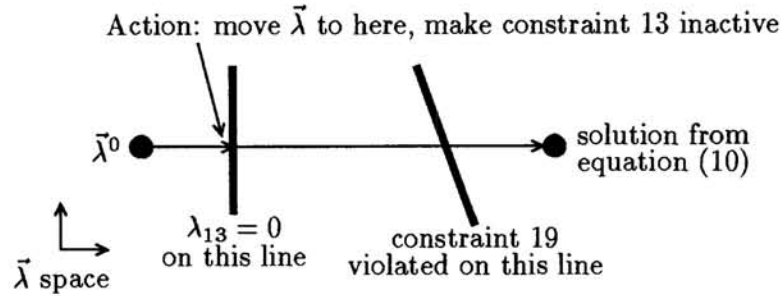

Figure 4: The position along the step where the constraints become violated or the Lagrange multipliers become zero can be computed analytically. The algorithm then takes the largest possible step without violating constraints or having the Lagrange multipliers become zero.

set if the inequality constraint is violated. A constraint can be moved off the active set if its Lagrange multiplier has changed sign[1].

Each step of the active set method attempts to adjust the vector of Lagrange multipliers to the values provided by equation (10). Let us parameterize the step from the old to the new Lagrange multipliers via a parameter $\alpha$:

$$\vec{\lambda} = \vec{\lambda}^0 + \alpha \Delta \vec{\lambda} \tag{11}$$

where $\vec{\lambda}^0$ is the vector of Lagrange multipliers before the step, $\Delta\vec{\lambda}$ is the step, and when $\alpha = 1$, the step is completed. Now, the amount of constraint violation and the Lagrange multipliers are linear functions of this $\alpha$. Therefore, we can analytically derive the $\alpha$ at which a constraint is violated or a Lagrange multiplier changes sign (see figure 4). For currently inactive constraints, the $\alpha$ for constraint violation is:

$$\alpha_k = -\frac{C_k + \sum_j \lambda_j^0 \sum_i A_{ji} A_{ki}}{\sum_j \Delta\lambda_j \sum_i A_{ji} A_{ki}} \tag{12}$$

For a currently active constraint, the $\alpha$ for a Lagrange multiplier sign change is simply:

$$\alpha_k = -\frac{\lambda_k^0}{\Delta\lambda_k} \tag{13}$$

We choose the constraint that has a smallest positive $\alpha_k$. If the smallest $\alpha_k$ is greater than 1, then the system has found the solution, and the final weights are computed from the Lagrange multipliers at the end of the step. Otherwise, if the $k$th constraint is active, we make it inactive, and vice versa. We then set the Lagrange multipliers to be the interpolated values from equation (11) with $\alpha = \alpha_k$. We finally re-evaluate equation (10) with the updated active set[2].

When this optimization algorithm is applied to the E13B font, the templates that result are shown in figure 5. When applied to characters that obey the specification, the classifier is guaranteed to give a 2:1 margin between the correct peak and any false peak caused by the confluence of two partial characters. Each template has 1523 constraints and takes 7.5 minutes on a SPARC 2 to train. Back-propagation on the 128,000 training examples that are equivalent to the constraints would obviously require much more computation time.

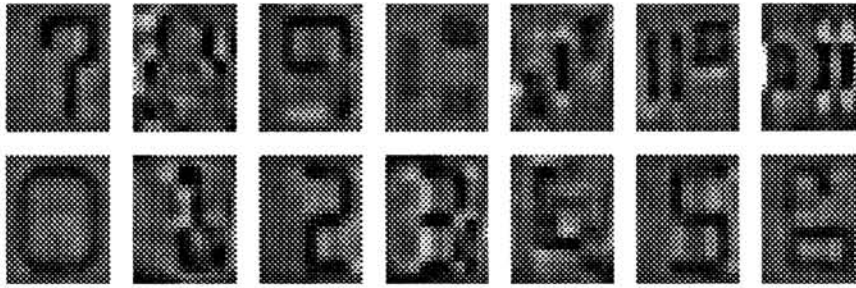

Figure 5: The weights for the fourteen E13B templates. The light pixels correspond to positive weights, while the dark pixels correspond to negative weights.

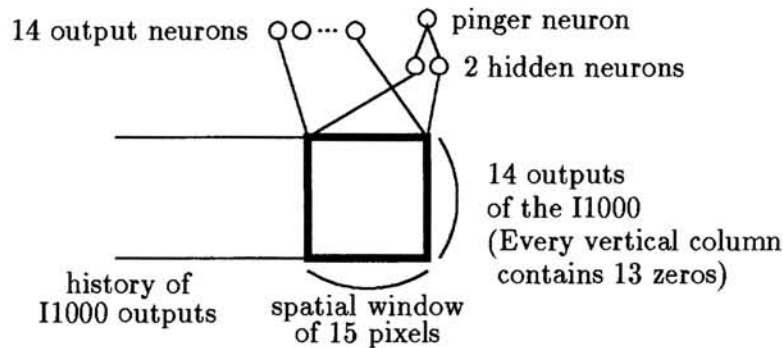

Figure 6: The software second layer

## 4    THE SOFTWARE SECOND LAYER

As a test of the linear classifier, we fabricated the I1000 and tested it with E13B characters on real checks. The system worked when the printing on the check obeyed the contrast specification of the font. However, some check printing companies use very light or very dark printing. Therefore, there was no single threshold that could consistently read the lightly printed checks without hallucinating characters on the dark checks. The retina shown in figure 2 does not have automatic gain control (AGC). One solution would have been to refabricate the chip using an AGC retina. However, we opted for a simpler solution.

The output of the I1000 chip is a 2-bit confidence level and a character code that is sent to an inexpensive microprocessor every 50 microseconds. Because this output bandwidth is low, it is feasible to put a small software second layer into this microprocessor to post-process and clean up the output of the I1000.

The architecture of this software second layer is shown in figure 6. The input to the second layer is a linearly time-warped history of the output of the I1000 chip. The time warping makes the second layer immune to changes in the velocity of the check in the slot. There is one output neuron that is a "pinger." That is, it is trained to turn on when the input to the I1000 chip is centered over any character (Platt, et al., 1992) (Martin & Pittman, 1992). There are fourteen other neurons that each correspond to a character in the font. These neurons are trained to turn on when the appropriate character is centered in the field, and otherwise turn off. The classification output is the output of the fourteen neurons only when the pinger neuron is on. Thus, the pinger neuron aids in segmentation.

Considering the entire network spanning both the hardware first layer and software

second layer, we have constructed a non-standard TDNN (Waibel, et. al., 1989) which recognizes characters.

We trained the second layer using standard back-propagation, with a training set gathered from real checks. Because the I1000 output bandwidth is quite low, collecting the data and training the network was not onerous. The second layer was trained on a data set of approximately 1000 real checks.

## 5   OVERALL PERFORMANCE

When the hardware first layer in the I1000 is combined with the software second layer, the performance of the system on real checks is quite impressive. We gathered a test set of 1500 real checks from across the country. This test set contained a variety of light and dark checks with unusual backgrounds. We swiped this test set through one system. Out of the 1500 test checks, the system only failed to read 2, due to staple holes in important locations of certain characters. As such, this test yielded a 99.995% character accuracy on real data.

## 6   CONCLUSIONS

For the I1000 analog VLSI OCR chip, we have created an effective hardware linear classifier that recognizes the E13B font. The behavior of this classifier was specified using constrained optimization. The classifier was designed to have a predictable margin of classification, be immune to lighting variations, and be resistant to random input noise. The classifier was trained using the active set method, which is an enhancement of Vapnik's separating hyperplane algorithm. We used the active set method to find the weights of a template in 7.5 minutes of SPARC 2 time, instead of training on a data set with 128,000 examples. To make the overall system resistant to contrast variation, we separately trained a software second layer on top of this first hardware layer, thereby constructing a non-standard TDNN.

The application discussed in this paper shows the utility of using the active set method to very rapidly create either a stand-alone linear classifier or a first layer of a multi-layer network.

## Footnotes

[1]The sign of the Lagrange multiplier indicates on which side of the inequality constraint the constrained minimum lies.

[2]For more details on active set methods, such as how to recognize infeasible constraints, consult (Gill, et al., 1981).

### References

P. Gill, W. Murray, M. Wright (1981), *Practical Optimization,* Section 5.2, Academic Press.

J. Lazzaro, S. Ryckebusch, M. Mahowald, C. Mead (1989), "Winner-Take-All Networks of $O(N)$ Complexity," *Advances in Neural Information Processing Systems,* **1**, D. Touretzky, ed., Morgan-Kaufmann, San Mateo, CA.

G. Martin, M. Rashid (1992), "Recognizing Overlapping Hand-Printed Characters by Centered-Object Integrated Segmentation and Recognition," *Advances in Neural Information Processing Systems,* **4**, Moody, J., Hanson, S., Lippmann, R., eds., Morgan-Kaufmann, San Mateo, CA.

J. Platt, J. Decker, and J. LeMoncheck (1992), Convolutional Neural Networks for the Combined Segmentation and Recognition of Machine Printed Characters, *USPS 5th Advanced Technology Conference,* **2**, 701-713.

V. Vapnik (1982), *Estimation of Dependencies Based on Empirical Data,* Addendum I, Section 2, Springer-Verlag.

A. Waibel, T. Hanazawa, G. Hinton, K. Shikano, K. Lang (1989), "Phoneme Recognition Using Time-Delay Neural Networks," *IEEE Transactions on Acoustics, Speech, and Signal Processing,* vol. 37, pp. 328–339.
